# Support Vector Method for Function Approximation, Regression Estimation, and Signal Processing

**Vladimir Vapnik**
AT&T Research
101 Crawfords Corner
Holmdel, NJ 07733
vlad@research.att.com

**Steven E. Golowich**
Bell Laboratories
700 Mountain Ave.
Murray Hill, NJ 07974
golowich@bell-labs.com

**Alex Smola***
GMD First
Rudower Shausee 5
12489 Berlin
asm@big.att.com

## Abstract

The Support Vector (SV) method was recently proposed for estimating regressions, constructing multidimensional splines, and solving linear operator equations [Vapnik, 1995]. In this presentation we report results of applying the SV method to these problems.

## 1 Introduction

The Support Vector method is a universal tool for solving multidimensional function estimation problems. Initially it was designed to solve pattern recognition problems, where in order to find a decision rule with good generalization ability one selects some (small) subset of the training data, called the Support Vectors (SVs). Optimal separation of the SVs is equivalent to optimal separation the entire data.

This led to a new method of representing decision functions where the decision functions are a linear expansion on a basis whose elements are nonlinear functions parameterized by the SVs (we need one SV for each element of the basis). This type of function representation is especially useful for high dimensional input space: the number of free parameters in this representation is equal to the number of SVs but does not depend on the dimensionality of the space.

Later the SV method was extended to real-valued functions. This allows us to expand high-dimensional functions using a small basis constructed from SVs. This

novel type of function representation opens new opportunities for solving various problems of function approximation and estimation.

In this paper we demonstrate that using the SV technique one can solve problems that in classical techniques would require estimating a large number of free parameters. In particular we construct one and two dimensional splines with an arbitrary number of grid points. Using linear splines we approximate non-linear functions. We show that by reducing requirements on the accuracy of approximation, one decreases the number of SVs which leads to data compression. We also show that the SV technique is a useful tool for regression estimation. Lastly we demonstrate that using the SV function representation for solving inverse ill-posed problems provides an additional opportunity for regularization.

## 2  SV method for estimation of real functions

Let $x \in R^n$ and $y \in R^1$. Consider the following set of real functions: a vector $x$ is mapped into some a priori chosen Hilbert space, where we define functions that are linear in their parameters

$$y = f(x, w) = \sum_{i=1}^{\infty} w_i \phi_i(x), \qquad w = (w_1, ..., w_N, ...) \in \Omega \tag{1}$$

In [Vapnik, 1995] the following method for estimating functions in the set (1) based on training data $(x_1, y_1), ..., (x_\ell, y_\ell)$ was suggested: find the function that minimizes the following functional:

$$R(w) = \frac{1}{\ell} \sum_{i=1}^{\ell} |y_i - f(x_i, w)|_\varepsilon + \gamma(w, w), \tag{2}$$

where

$$|y - f(x, w)|_\varepsilon = \begin{cases} 0 & \text{if } |y - f(x, w)| < \varepsilon, \\ |y - f(x, w)| - \varepsilon & \text{otherwise,} \end{cases} \tag{3}$$

$(w, w)$ is the inner product of two vectors, and $\gamma$ is some constant. It was shown that the function minimizing this functional has a form:

$$f(x, \alpha, \alpha^*) = \sum_{i=1}^{\ell} (\alpha_i^* - \alpha_i)(\Phi(x_i), \Phi(x)) + b \tag{4}$$

where $\alpha_i^*, \alpha_i \geq 0$ with $\alpha_i^* \alpha_i = 0$ and $(\Phi(x_i), \Phi(x))$ is the inner product of two elements of Hilbert space.

To find the coefficients $\alpha_i^*$ and $\alpha_i$ one has to solve the following quadratic optimization problem: maximize the functional

$$W(\alpha^*, \alpha) = -\varepsilon \sum_{i=1}^{\ell} (\alpha_i^* + \alpha_i) + \sum_{i=1}^{\ell} y(\alpha_i^* - \alpha_i) - \frac{1}{2} \sum_{i,j=1}^{\ell} (\alpha_i^* - \alpha_i)(\alpha_j^* - \alpha_j)(\Phi(x_i), \Phi(x_j)),$$

$$\tag{5}$$

subject to constraints

$$\sum_{i=1}^{\ell} (\alpha_i^* - \alpha_i) = 0, \quad 0 \leq \alpha_i, \alpha_i^* \leq C, \quad i = 1, ..., \ell. \tag{6}$$

The important feature of the solution (4) of this optimization problem is that only some of the coefficients $(\alpha_i^* - \alpha_i)$ differ from zero. The corresponding vectors $x_i$ are called Support Vectors (SVs). Therefore (4) describes an expansion on SVs.

It was shown in [Vapnik, 1995] that to evaluate the inner products $(\Phi(x_i), \Phi(x))$ both in expansion (4) and in the objective function (5) one can use the general form of the inner product in Hilbert space. According to Hilbert space theory, to guarantee that a symmetric function $K(u, v)$ has an expansion

$$K(u, v) = \sum_{k=1}^{\infty} a_k \phi_k(u) \phi_k(v)$$

with positive coefficients $a_k > 0$, i.e. to guarantee that $K(u, v)$ is an inner product in some feature space $\Phi$, it is necessary and sufficient that the conditions

$$\int K(u, v) g(u) g(v) \, du \, dv > 0 \tag{7}$$

be valid for any non-zero function $g$ on the Hilbert space (Mercer's theorem).

Therefore, in the SV method, one can replace (4) with

$$f(x, \alpha, \alpha^*) = \sum_{i=1}^{\ell} (\alpha_i^* - \alpha_i) K(x, x_i) + b \tag{8}$$

where the inner product $(\Phi(x_i), \Phi(x))$ is defined through a kernel $K(x_i, x)$. To find coefficients $\alpha_i^*$ and $\alpha_i$ one has to maximize the function

$$W(\alpha^*, \alpha) = -\varepsilon \sum_{i=1}^{\ell} (\alpha_i^* + \alpha_i) + \sum_{i=1}^{\ell} y(\alpha_i^* - \alpha_i) - \frac{1}{2} \sum_{i,j=1}^{\ell} (\alpha_i^* - \alpha_i)(\alpha_j^* - \alpha_j) K(x_i, x_j) \tag{9}$$

subject to constraints (6).

## 3 Constructing kernels for inner products

To define a set of approximating functions one has to define a kernel $K(x_i, x)$ that generates the inner product in some feature space and solve the corresponding quadratic optimization problem.

### 3.1 Kernels generating splines

We start with the spline functions. According to their definition, splines are piecewise polynomial functions, which we will consider on the set $[0, 1]$. Splines of order $n$ have the following representation

$$f_n(x) = \sum_{r=0}^{n} a_r x^r + \sum_{s=1}^{N} w_j (x - t_s)_+^n \tag{10}$$

where $(x - t)_+ = \max\{(x - t), \ 0\}$, $t_1, ..., t_N \in [0, 1]$ are the nodes, and $a_r, w_j$ are real values. One can consider the spline function (10) as a linear function in the $n + N + 1$ dimensional feature space spanned by

$$1, x, ..., x^n, (x - t_1)_+^n, ..., (x - t_N)_+^n.$$

Therefore the inner product that generates splines of order $n$ in one dimension is

$$K(x_i, x_j) = \sum_{r=0}^{n} x_i^r x_j^r + \sum_{s=1}^{N} (x_i - t_s)_+^n (x_j - t_s)_+^n. \tag{11}$$

Two dimensional splines are linear functions in the $(N + n + 1)^2$ dimensional space

$$1, x, ..., x^n, y, ..., y^n, ..., (x - t_1)_+^n (y - t_1^*)_+^n, ..., (x - t_N)_+^n (y - t_N^*)_+^n. \tag{12}$$

Let us denote by $u_i = (x_i, y_i)$, $u_j = (x_i, y_j)$ two two-dimensional vectors. Then the generating kernel for two dimensional spline functions of order $n$ is

$$K(u_i, u_j) = K(x_i, x_j) K(y_i, y_j)$$

It is easy to check that the generating kernel for the $m$-dimensional splines is the product of $m$ one-dimensional generating kernels.

In applications of the SV method the number of nodes does not play an important role. Therefore, we introduce splines of order $d$ with an infinite number of nodes $S_d^{(\infty)}$. To do this in the $R^1$ case, we map any real value $x_i$ to the element $1, x_i, ..., x_i^n, (x_i - t)_+^n$ of the Hilbert space. The inner product becomes

$$K(x_i, x_j) = \sum_{r=0}^{n} x_i^r x_j^r + \int_0^1 (x_i - t)_+^n (x_j - t)_+^n dt \tag{13}$$

For linear splines $S_1^{(\infty)}$ we therefore have the following generating kernel:

$$K(x_i, x_j) = 1 + x_i x_j + x_i x_j \min(x_i, x_j) - \frac{(x_i + x_j)}{2} (\min(x_i, x_j))^2 + \frac{(\min(x_i, x_j))^3}{3}. \tag{14}$$

In many applications expansions in $B_n$-splines [Unser & Aldroubi, 1992] are used, where

$$B_n(x) = \sum_{r=0}^{n+1} \frac{(-1)^r}{n!} \binom{n+1}{r} \left( x + \frac{n+1}{2} - r \right)_+^n.$$

One may use $B_n$-splines to perform a construction similar to the above, yielding the kernel

$$K(x_i, x_j) = \int_{-\infty}^{\infty} B_n(x_i - t) B_n(x_j - t) dt = B_{2n+1}(x_i - x_j).$$

## 3.2    Kernels generating Fourier expansions

Lastly, Fourier expansion can be considered as a hyperplane in following $2N + 1$ dimensional feature space

$$\frac{1}{\sqrt{2}}, \cos x, \sin x, ..., \cos Nx, \sin Nx.$$

The inner product in this space is defined by the Dirichlet formula:

$$K(x_i, x_j) = \frac{1}{2} + \sum_{r=1}^{N} (\cos rx_i \cos rx_j + \sin rx_i \sin rx_j) = \frac{\sin(N + 1/2)(x_i - x_j)}{\sin \frac{x_i - x_j}{2}}. \tag{15}$$

# 4  Function estimation and data compression

In this section we approximate functions on the basis of observations at $\ell$ points

$$(x_1, y_1), ..., (x_\ell, y_\ell). \tag{16}$$

We demonstrate that to construct an approximation within an accuracy of $\pm\varepsilon$ at the data points, one can use only the subsequence of the data containing the SVs.

We consider approximating the one and two dimensional functions

$$f(x) = \text{sinc}|x| = \frac{\sin|x|}{|x|} \tag{17}$$

on the basis of a sequence of measurements (without noise) on the uniform lattice (100 for the one dimensional case and 2,500 for the two-dimensional case).

For different $\varepsilon$ we approximate this function by linear splines from $S_1^{(\infty)}$.

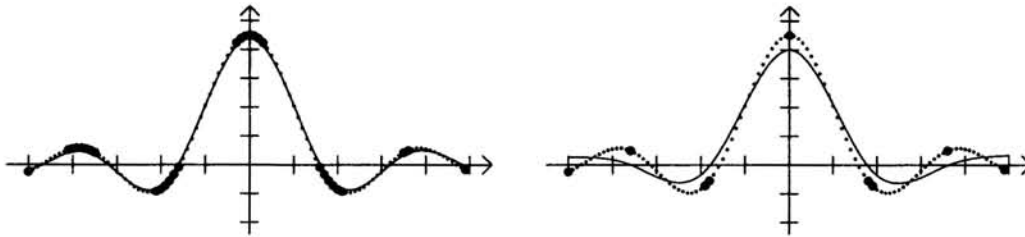

Figure 1: Approximations with different levels of accuracy require different numbers of SV: 31 SV for $\varepsilon = 0.02$ (left) and 9 SV for $\varepsilon = 0.1$ (right). Large dots indicate SVs.

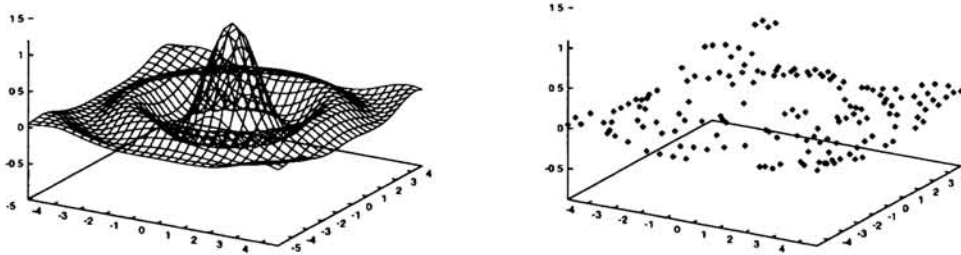

Figure 2: Approximation of $f(x, y) = \text{sinc}\sqrt{x^2 + y^2}$ by two dimensional linear splines with accuracy $\varepsilon = 0.01$ (left) required 157 SV (right)

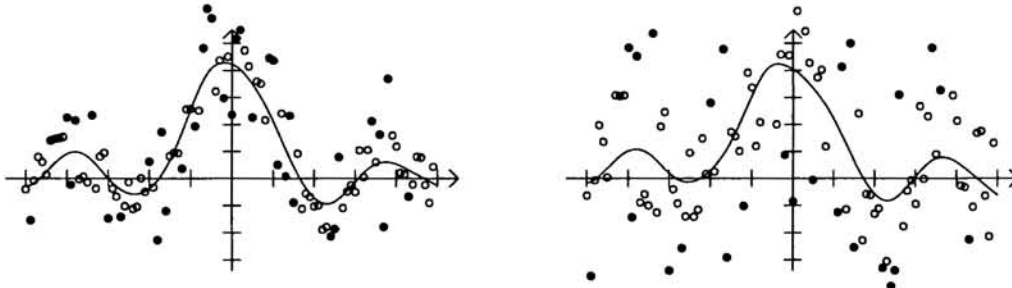

Figure 3: sinc$x$ function corrupted by different levels of noise ($\sigma = 0.2$ left, 0.5 right) and its regression. Black dots indicate SV, circles non-SV data.

## 5   Solution of the linear operator equations

In this section we consider the problem of solving linear equations in the set of functions defined by SVs. Consider the problem of solving a linear operator equation

$$Af(t) = F(x), \qquad f(t) \in \Xi, \ F(x) \in \Psi, \tag{18}$$

where we are given measurements of the right hand side

$$(x_1, F_1), ..., (x_\ell, F_\ell). \tag{19}$$

Consider the set of functions $f(t, w) \in \Xi$ linear in some feature space $\{\Phi(t) = (\phi_0(t), ..., \phi_N(t), ...)\}$:

$$f(t, w) = \sum_{r=0}^{\infty} w_r \phi_r(t) = (W, \Phi(t)). \tag{20}$$

The operator $A$ maps this set of functions into

$$F(x, w) = Af(t, w) = \sum_{r=0}^{\infty} w_r A\phi_r(t) = \sum_{r=0}^{\infty} w_r \psi_r(x) = (W, \Psi(x)) \tag{21}$$

where $\psi_r(x) = A\phi_r(t)$, $\Psi(x) = (\psi_1(x), ..., \psi_N(x), ...)$. Let us define the generating kernel in image space

$$K(x_i, x_j) = \sum_{r=0}^{\infty} \psi_r(x_i)\psi_r(x_j) = (\Psi(x_i), \Psi(x_j)) \tag{22}$$

and the corresponding cross-kernel function

$$\mathcal{K}(x_i, t) = \sum_{r=0}^{\infty} \psi_r(x_i)\phi_r(t) = (\Psi(x_i), \Phi(t)). \tag{23}$$

The problem of solving (18) in the set of functions $f(t, w) \in \Xi$ (finding the vector $W$) is equivalent to the problem of regression estimation (21) using data (19).

To estimate the regression on the basis of the kernel $K(x_i, x_j)$ one can use the methods described in Section 1. The obtained parameters $(\alpha_i^* - \alpha_i, \ i = 1, ...\ell)$ define the approximation to the solution of equation (18) based on data (19):

$$f(t, \alpha) = \sum_{i=1}^{\ell} (\alpha_i^* - \alpha_i)\mathcal{K}(x_i, t).$$

We have applied this method to solution of the Radon equation

$$\int_{-a(m)}^{a(m)} f(m \cos \mu + u \sin \mu, \ m \sin \mu - u \cos \mu) du = p(m, \mu),$$

$$-1 \le m \le 1, \ \ 0 < \mu < \pi, \ \ a(m) = \sqrt{1 - m^2} \tag{24}$$

using noisy observations $(m_1, \mu_1, p_1), ..., (m_\ell, \mu_\ell, p_\ell)$, where $p_i = p(m_i, \mu_i) + \xi_i$ and $\{\xi_i\}$ are independent with $E\xi_i = 0$, $E\xi_i^2 < \infty$.

For two-dimensional linear splines $S_1^{(\infty)}$ we obtained analytical expressions for the ꞏkernel (22) and cross-kernel (23). We have used these kernels for solving the corresponding regression problem and reconstructing images based on data that is similar to what one might get from a Positron Emission Tomography scan [Shepp, Vardi & Kaufman, 1985].

A remarkable feature of this solution is that it avoids a pixel representation of the function which would require the estimation of 10,000 to 60,000 parameters. The spline approximation shown here required only 172 SVs.

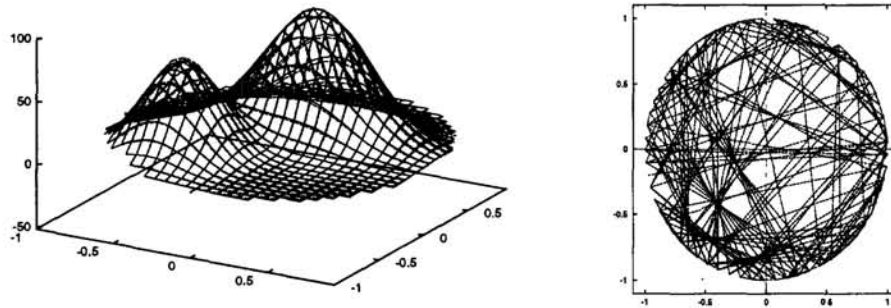

Figure 4: Original image (dashed line) and its reconstruction (solid line) from 2,048 observations (left). 172 SVs (support lines) were used in the reconstruction (right).

# 6    Conclusion

In this article we present a new method of function estimation that is especially useful for solving multi-dimensional problems. The complexity of the solution of the function estimation problem using the SV representation depends on the complexity of the desired solution (i.e. on the required number of SVs for a reasonable approximation of the desired function) rather than on the dimensionality of the space. Using the SV method one can solve various problems of function estimation both in statistics and in applied mathematics.

## Acknowledgments

We would like to thank Chris Burges (Lucent Technologies) and Bernhard Schölkopf (MPIK Tübingen) for help with the code and useful discussions.

This work was supported in part by NSF grant PHY 95-12729 (Steven Golowich) and by ARPA grant N00014-94-C-0186 and the German National Scholarship Foundation (Alex Smola).

## Footnotes

*smola@prosun.first.gmd.de

## References

1. Vladimir Vapnik, "The Nature of Statistical Learning Theory", 1995, Springer Verlag N.Y., 189 p.

2. Michael Unser and Akram Aldroubi, "Polynomial Splines and Wevelets - A Signal Perspectives", In the book: "Wavelets –A tutorial in Theory and Applications", C.K. Chui (ed) pp. 91 − 122, 1992 Academic Press, Inc.

3. L. Shepp, Y. Vardi, and L. Kaufman, "A statistical model for Positron Emission Tomography," *J. Amer. Stat. Assoc.* **80:389** pp. 8-37 1985.